# Measuring model complexity with the prior predictive

**Wolf Vanpaemel** *
Department of Psychology
University of Leuven
Belgium.
`wolf.vanpaemel@psy.kuleuven.be`

## Abstract

In the last few decades, model complexity has received a lot of press. While many methods have been proposed that jointly measure a model's descriptive adequacy and its complexity, few measures exist that measure complexity in itself. Moreover, existing measures ignore the parameter prior, which is an inherent part of the model and affects the complexity. This paper presents a stand alone measure for model complexity, that takes the number of parameters, the functional form, the range of the parameters and the parameter prior into account. This Prior Predictive Complexity (PPC) is an intuitive and easy to compute measure. It starts from the observation that model complexity is the property of the model that enables it to fit a wide range of outcomes. The PPC then measures how wide this range exactly is.

**keywords:** Model Selection & Structure Learning; Model Comparison Methods; Perception

The recent revolution in model selection methods in the cognitive sciences was driven to a large extent by the observation that computational models can differ in their complexity. Differences in complexity put models on unequal footing when their ability to approximate empirical data is assessed. Therefore, models should be penalized for their complexity when their adequacy is measured. The balance between descriptive adequacy and complexity has been termed generalizability [1, 2].

Much attention has been devoted to developing, advocating, and comparing different measures of generalizability (for a recent overview, see [3]). In contrast, measures of complexity have received relatively little attention. The aim of the current paper is to propose and illustrate a stand alone measure of model complexity, called the Prior Predictive Complexity (PPC). The PPC is based on the intuitive idea that a complex model can predict many outcomes and a simple model can predict a few outcomes only.

First, I discuss existing approaches to measuring model complexity and note some of their limitations. In particular, I argue that currently existing measures ignore one important aspect of a model: the prior distribution it assumes over the parameters. I then introduce the PPC, which, unlike the existing measures, is sensitive to the parameter prior. Next, the PPC is illustrated by calculating the complexities of two popular models of information integration.

## 1 Previous approaches to measuring model complexity

A first approach to assess the (relative) complexity of models relies on simulated data. Simulation-based methods differ in how these artificial data are generated. A first, atheoretical approach uses random data [4, 5]. In the semi-theoretical approach, the data are generated from some theoretically

interesting functions, such as the exponential or the logistic function [4]. Using these approaches, the models under consideration are equally complex if each model provides the best optimal fit to roughly the same number of data sets. A final approach to generating artificial data is a theoretical one, in which the data are generated from the models of interest themselves [6, 7]. The parameter sets used in the generation can either be hand-picked by the researcher, estimated from empirical data or drawn from a previously specified distribution. If the models under consideration are equally complex, each model should provide the best optimal fit to self-generated data more often than the other models under consideration do.

One problem with this simulation-based approach is that it is very labor intensive. It requires generating a large amount of artificial data sets, and fitting the models to all these data sets. Further, it relies on choices that are often made in an arbitrary fashion that nonetheless bias the results. For example, in the semi-theoretical approach, a crucial choice is which functions to use. Similarly, in the theoretical approach, results are heavily influenced by the parameter values used in generating the data. If they are fixed, on what basis? If they are estimated from empirical data, from which data? If they are drawn randomly, from which distribution? Further, a simulation study only gives a rough idea of complexity differences but provides no direct measure reflecting the complexity.

A number of proposals have been made to measure model complexity more directly. Consider a model $M$ with $k$ parameters, summarized in the parameter vector $\theta = (\theta_1, \theta_2, \ldots, \theta_k,)$ which has a range indicated by $\Omega$. Let $d$ denote the data and $p(d|\theta, M)$ the likelihood. The most straightforward measure of model complexity is the parametric complexity (PC), which simply counts the number of parameters:

$$\text{PC} = k. \tag{1}$$

PC is attractive as a measure of model complexity since it is very easy to calculate. Further, it has a direct and well understood relation toward complexity: the more parameters, the more complex the model. It is included as the complexity term of several generalizability measures such as AIC [8] and BIC [9], and it is at the heart of the Likelihood Ratio Test.

Despite this intuitive appeal, PC is not free from problems. One problem with PC is that it reflects only a single aspect of complexity. Also the parameter range and the functional form (the way the parameters are combined in the model equation) influence a model's complexity, but these dimensions of complexity are ignored in PC [2, 6].

A complexity measure that takes these three dimensions into account is provided by the geometric complexity (GC) measure, which is inspired by differential geometry [10]. In GC, complexity is conceptualized as the number of distinguishable probability distributions a model can generate. It is defined by

$$\text{GC} = \frac{k}{2} \ln \frac{n}{2\pi} + \ln \int_\Omega \sqrt{\det I(\theta|M)} d\theta, \tag{2}$$

where $n$ indicates the size of the data sample and $I(\theta)$ is the Fisher Information Matrix:

$$I_{ij}(\theta|M) = -E_\theta \frac{\partial^2 \ln p(d|\theta, M)}{\partial \theta_i \partial \theta_j}. \tag{3}$$

Note that $I(\theta|M)$ is determined by the likelihood function $p(d|\theta, M)$, which is in turn determined by the model equation. Hence GC is sensitive to the number of parameters (through $k$), the functional form (through $I$), and the range (through $\Omega$). Quite surprisingly, GC turns out to be equal to the complexity term used in one version of Minimum Description Length (MDL), a measure of generalizability developed within the domain of information theory [2, 11, 12, 13].

GC contrasts favorably with PC, in the sense that it takes three dimensions of complexity into account rather than a single one. A major drawback of GC is that, unlike PC, it requires considerable technical sophistication to be computed, as it relies on the second derivative of the likelihood. A more important limitation of both PC and GC is that these measures are insensitive to yet another important dimension contributing to model complexity: the prior distribution over the model parameters. The relation between the parameter prior distribution and model complexity is discussed next.

## 2 Model complexity and the parameter prior

The growing popularity of Bayesian methods in psychology has not only raised awareness that model complexity should be taken into account when testing models [6], it has also drawn attention to the fact that in many occasions, relevant prior information is available [14]. In Bayesian methods, there is room to incorporate this information in two different flavors: as a prior distribution over the models, or as a prior distribution over the parameters. Specifying a model prior is a daunting task, so almost invariably, the model prior is taken to be uniform (but see [15] for an exception). In contrast, information regarding the parameter is much easier to include, although still challenging (e.g., [16]).

There are two ways to formalize prior information about a model's parameters: using the parameter prior range (often referred to as simply the range) and using the parameter prior distribution (often referred to as simply the prior). The prior range indicates which parameter values are allowed and which are forbidden. The prior distribution indicates which parameter values are likely and which are unlikely. Models that share the same equation and the same range but differ in the prior distribution can be considered different models (or at least different model versions), just like models that share the same equation but differ in range are different model versions. Like the parameter prior range, the parameter prior distribution influences the model complexity. In general, a model with a vague parameter prior distribution is more complex than a model with a sharply peaked parameter prior distribution, much as a model with a broad-ranged parameter is more complex than the same model where the parameter is heavily restricted.

To drive home the point that the parameter prior should be considered when model complexity is assessed, consider the following "fair coin" model $M_f$ and a "biased coin" model $M_b$. There is a clear intuitive complexity difference between these models: $M_b$ is more complex than $M_f$. The most straightforward way to formalize these models is as follows, where $p_h$ denotes the probability of observing heads:

$$p_h = 1/2, \tag{4}$$

for model $M_f$ and the triplet of equations

$$p_h = \theta \tag{5}$$
$$0 \leq \theta \leq 1$$
$$p(\theta) = 1,$$

jointly define model $M_b$. The range forbids values smaller than 0 or greater than 1 because $p_h$ is a proportion. As $M_f$ and $M_b$ have a different number of parameters, both PC and GC, being sensitive to the number of parameters, pick up the difference in model complexity between the models.

Alternatively, model $M_f$ could be defined as follows:

$$p_h = \theta \tag{6}$$
$$0 \leq \theta \leq 1$$
$$p(\theta) = \delta(\theta - \frac{1}{2}),$$

where $\delta(x)$ is the Dirac delta. Note that the model formalized in Equation 6 is exactly identical the model formalized in Equation 4. However, relying on the formulation of model $M_f$ in Equation 6, PC and GC now judge $M_f$ and $M_b$ to be equally complex: both models share the same model equation (which implies they have the same number of parameters and the same functional form) and the same range for the parameter. Hence, PC and GC make an incorrect judgement of the complexity difference between both models. This misjudgement is a direct result of the insensitivity of these measures to the parameter prior. As models $M_f$ and $M_b$ have different prior distributions over their parameter, a measure sensitive to the prior would pick up the complexity difference between these models. Such a measure is introduced next.

## 3 The Prior Predictive Complexity

Model complexity refers to the property of the model that enables it to predict a wide range of data patterns [2]. The idea of the PPC is to measure how wide this range exactly is. A complex model

can predict many outcomes, and a simple model can predict a few outcomes only. Model simplicity, then, refers to the property of placing restrictions on the possible outcomes: the greater restrictions, the greater the simplicity.

To understand how model complexity is measured in the PPC, it is useful to think about the universal interval (UI) and the predicted interval (PI). The universal interval is the range of outcomes that could potentially be observed, irrespective of any model. For example, in an experiment with $n$ binomial trials, it is impossible to observe less that zero successes, or more than $n$ successes, so the range of possible outcomes is $[0, n]$. Similarly, the universal interval for a proportion is $[0, 1]$. The predicted interval is the interval containing all outcomes the model predicts.

An intuitive way to gauge model complexity is then the cardinality of the predicted interval, relative to the cardinality of the universal interval, averaged over all $m$ conditions or stimuli:

$$\text{PPC} = \frac{1}{m} \sum_{i=1}^{m} \frac{|\text{PI}_i|}{|\text{UI}_i|}. \tag{7}$$

A key aspect of the PPC is deriving the predicted interval. For a parameterized likelihood-based model, prediction takes the form of a distribution over all possible outcomes for some future, yet-to-be-observed data $d$ under some model $M$. This distribution is called the prior predictive distribution (ppd) and can be calculated using the law of total probability:

$$p(d|M) = \int_{\Omega} p(d|\theta, M)p(\theta|M)\mathrm{d}\theta. \tag{8}$$

Predicting the probability of unseen future data $d$ arising under the assumption that model $M$ is true involves integrating the probability of the data for each of the possible parameter values, $p(d|\theta, M)$, as weighted by the prior probability of each of these values, $p(\theta|M)$.

Note that the ppd relies on the number of parameters (through the number of integrals and the likelihood), the model equation (through the likelihood), and the parameter range (through $\Omega$). Therefore, as GC, the PPC is sensitive to all these aspects. In contrast to GC, however, the ppd, and hence the PPC, also relies on the parameter prior.

Since predictions are made probabilistically, virtually all outcomes will be assigned some prior weight. This implies that, in principle, the predicted interval equals the universal interval. However, for some outcomes the assigned weight will be extremely small. Therefore, it seems reasonable to restrict the predicted interval to the smallest interval that includes some predetermined amount of the prior mass. For example, the 95% predictive interval is defined by those outcomes with the highest prior mass that together make up 95% of the prior mass.

Analytical solutions to the integral defining the ppd are rarely available. Instead, one should rely on approximations to the ppd by drawing samples from it. In the current study, sampling was performed using WinBUGS [17, 18], a highly versatile, user friendly, and freely available software package. It contains sophisticated and relatively general-purpose Markov Chain Monte Carlo (MCMC) algorithms to sample from any distribution of interest.

## 4 An application example

The PPC is illustrated by comparing the complexity of two popular models of information integration, which attempt to account for how people merge potentially ambiguous or conflicting information from various sensorial sources to create subjective experience. These models either assume that the sources of information are combined additively (the Linear Integration Model; LIM; [19]) or multiplicatively (the Fuzzy Logical Model of Perception; FLMP; [20, 21]).

### 4.1 Information integration tasks

A typical information integration task exposes participants simultaneously to different sources of information and requires this combined experience to be identified in a forced-choice identification task. The presented stimuli are generated from a factorial manipulation of the sources of information by systematically varying the ambiguity of each of the sources. The relevant empirical data consist

of, for each of the presented stimuli, the counts $k_m$ of the number of times the $m$th stimulus was identified as one of the response alternatives, out of the $t_m$ trials on which it was presented.

For example, an experiment in phonemic identification could involve two phonemes to be identified, $/ba/$ and $/da/$ and two sources of information, auditory and visual. Stimuli are created by crossing different levels of audible speech, varying between $/ba/$ and $/da/$, with different levels of visible speech, also varying between these alternatives. The resulting set of stimuli spans a continuum between the two syllables. The participant is then asked to listen and to watch the speaker, and based on this combined audiovisual experience, to identify the syllable as being either $/ba/$ or $/da/$. In the so-called *expanded factorial design*, not only bimodal stimuli (containing both auditory and visual information) but also unimodal stimuli (providing only a single source of information) are presented.

## 4.2 Information integration models

In what follows, the formal description of the LIM and the FLMP is outlined for a design with two response alternatives ($/da/$ or $/ba/$) and two sources (auditory and visual), with $I$ and $J$ levels, respectively. In such a two-choice identification task, the counts $k_m$ follow a Binomial distribution:

$$k_m \sim \text{Binomial}(p_m, t_m), \tag{9}$$

where $p_m$ indicates the probability that the $m$th stimulus is identified as $/da/$.

### 4.2.1 Model equation

The probability for the stimulus constructed with the $i$th level of the first source and the $j$th level of the second being identified as $/da/$ is computed according to the choice rule:

$$p_{ij} = \frac{s\,(ij, /da/)}{s\,(ij, /da/) + s\,(ij, /ba/)}, \tag{10}$$

where $s\,(ij, /da/)$ represents the overall degree of support for the stimulus to be $/da/$.

The sources of information are assumed to be evaluated independently, implying that different parameters are used for the different modalities. In the present example, the degree of auditory support for $/da/$ is denoted by $a_i$ ($i = 1, \ldots, I$) and the degree of visual support for $/da/$ by $b_j$ ($j = 1, \ldots, J$).

When a unimodal stimulus is presented, the overall degree of support for each alternative is given by $s\,(i*, /da/) = a_i$ and $s\,(*j, /da/) = b_j$, where the asterisk (*) indicates the absence of information, implying that Equation 10 reduces to

$$p_{i*} = a_i \quad \text{and} \quad p_{*j} = b_j. \tag{11}$$

When a bimodal stimulus is presented, the overall degree of support for each alternative is based on the integration or blending of both these sources. Hence, for bimodal stimuli, $s\,(ij, /da/) = a_i \otimes b_j$, where the operator $\otimes$ denotes the combination of both sources. Hence, Equation 10 reduces to

$$p_{ij} = \frac{a_i \otimes b_j}{a_i \otimes b_j + (1 - a_i) \otimes (1 - b_j)}. \tag{12}$$

The LIM assumes an additive combination, i.e., $\otimes = +$, so Equation 12 becomes

$$p_{ij} = \frac{a_i + b_j}{2}. \tag{13}$$

The FLMP, in contrast, assumes a multiplicative combination, i.e., $\otimes = \times$, so Equation 12 becomes

$$p_{ij} = \frac{a_i b_j}{a_i b_j + (1 - a_i)(1 - b_j)}. \tag{14}$$

### 4.2.2 Parameter prior range and distribution

Each level of auditory and visual support for $/da/$ (i.e., $a_i$ and $b_j$, respectively) is associated with a free parameter, which implies that the FLMP and the LIM have an equal number of free parameters, $I + J$. Each of these parameters is constrained to satisfy $0 \leq a_i, b_j \leq 1$.

The original formulations of the LIM and FLMP unfortunately left the parameter priors unspecified. However, an implicit assumption that has been commonly used is a uniform prior for each of the parameters. This assumption implicitly underlies classical and widely adopted methods for model evaluation using accounted percentage of variance or maximum likelihood.

$$a_i \sim \text{Uniform}(0,1) \quad \text{and} \quad b_i \sim \text{Uniform}(0,1) \quad \text{for} \quad i = 1, \ldots, I; j = 1, \ldots, J. \quad (15)$$

The models relying on this set of uniform priors will be referred to as $\text{LIM}_u$ and $\text{FLMP}_u$.

Note that $\text{LIM}_u$ and $\text{FLMP}_u$ treat the different parameters as independent. This approach misses important information. In particular, the experimental design is such that the amount of support for each level $i + 1$ is always higher than for level $i$. Because parameter $a_i$ (or $b_i$) corresponds to the degree of auditory (or visual) support for a unimodal stimulus at the $i$th level, it seems reasonable to expect the following orderings among the parameters to hold (see also [6]):

$$a_j > a_i \quad \text{and} \quad b_j > b_i \quad \text{for} \quad j > i. \quad (16)$$

The models relying on this set of ordered priors will be referred to as $\text{LIM}_o$ and $\text{FLMP}_o$.

### 4.3 Complexity and experimental design

It is tempting to consider model complexity as an inherent characteristic of a model. For some models and for some measures of complexity this is clearly the case. Consider, for example, model $M_b$. In any experimental design (i.e., a number of coin tosses), $\text{PC}_{M_b} = 1$. However, more generally, this is not the case. Focusing on the FLMP and the LIM, it is clear that even a simple measure as PC depends crucially on (some aspects of) the experimental design. In particular, every level corresponds to a new parameter, so $\text{PC} = I + J$. Similarly, GC is dependent on design choices. The PPC is not different in this respect.

The design sensitivity implies that one can only make sensible conclusions about differences in model complexity by using different designs. In an information integration task, the design decisions include the type of design (expanded or not), the number of sources, the number of response alternatives, the number of levels for each source, and the number of observations for each stimulus (sample size). The present study focuses on the expanded factorial designs with two sources and two response alternatives. The additional design features were varied: both a $5 \times 5$ and a $8 \times 2$ design were considered, using three different sample sizes (20, 60 and 150, following [2]).

### 4.4 Results

Figure 1 shows the 99% predicted interval in the $8 \times 2$ design with $n = 150$. Each panel corresponds to a different model. In each panel, each of the 26 stimuli is displayed on the x-axis. The first eight stimuli correspond to the stimuli with the lowest level of visual support, and are ordered in increasing order of auditory support. The next eight stimuli correspond to the stimuli with the highest level of visual support. The next eight stimuli correspond to the unimodal stimuli where only auditory information is provided (again ranked in increasing order). The final two stimuli are the unimodal visual stimuli.

Panel A shows that the predicted interval of $\text{LIM}_u$ nearly equals the universal interval, ranging between 0 and 1. This indicates that almost all outcomes are given a non-negligible prior mass by $\text{LIM}_u$, making it almost maximally complex. $\text{FLMP}_u$ is even more complex. The predicted interval, shown in Panel B, virtually equals the universal interval, indicating that the model predicts virtually every possible outcome. Panels C and D show the dramatic effect of incorporating relevant prior information in the models. The predicted intervals of both $\text{LIM}_o$ and $\text{FLMP}_o$ are much smaller than their counterparts using the uniform priors.

Focusing on the comparison between LIM and FLMP, the PPC indicates that the latter is more complex than the former. This observation holds irrespective of the model version (assuming uniform

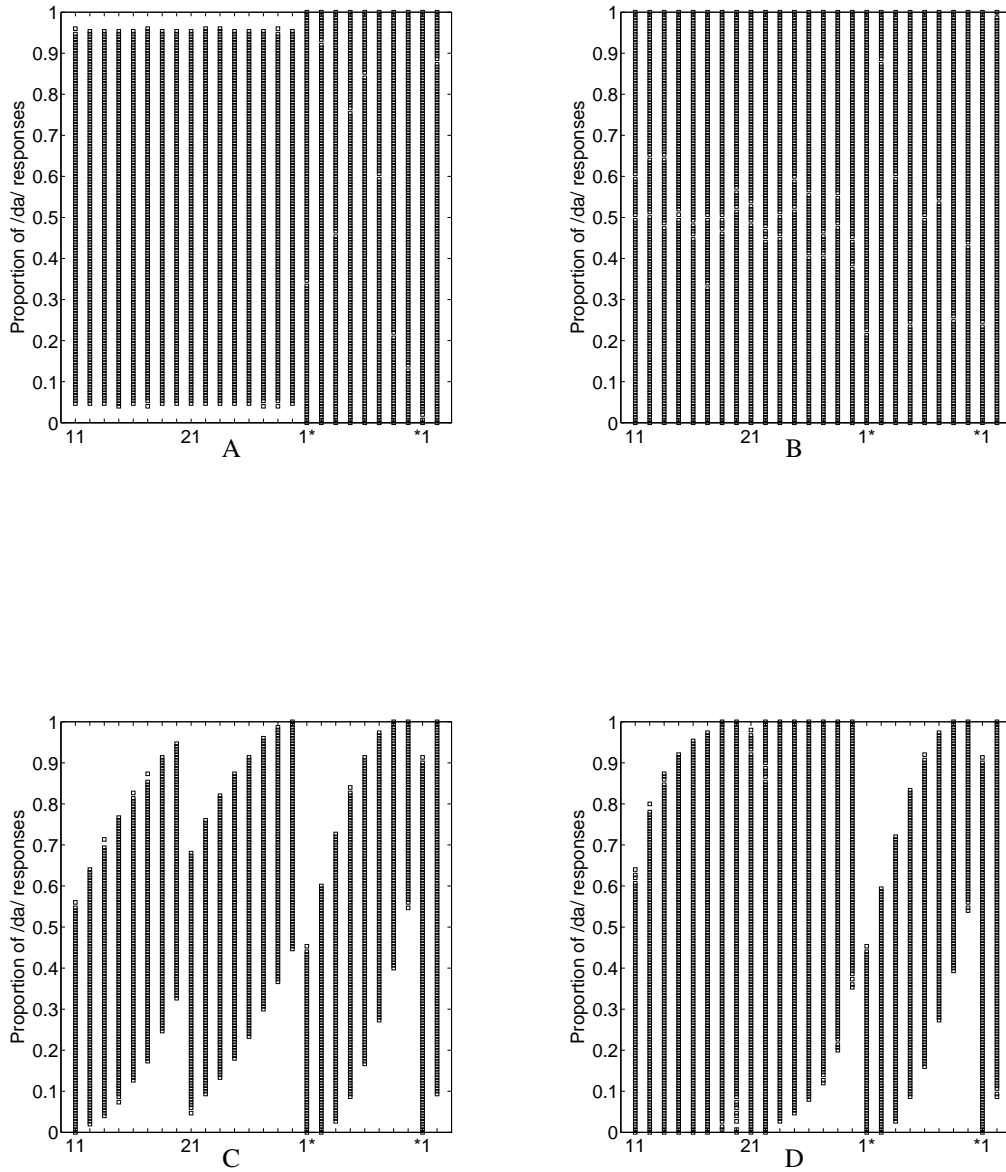

Figure 1: The 99% predicted interval for each of the 26 stimuli (x-axis) according to $LIM_u$ (Panel A), $FLMP_u$ (Panel B), $LIM_o$ (Panel C), and $FLMP_o$ (Panel D).

Table 1: PPC, based on the 99% predicted interval, for four models across six different designs.

| | $5 \times 5$ | | | $8 \times 2$ | | |
|---|---|---|---|---|---|---|
| | 20 | 60 | 150 | 20 | 60 | 150 |
| $LIM_u$ | 0.97 | 0.94 | 0.93 | .97 | 0.95 | 0.94 |
| $FLMP_u$ | 1 | 1 | 0.99 | 1 | 1 | 0.99 |
| $LIM_o$ | 0.75 | 0.67 | 0.64 | 0.77 | 0.69 | 0.66 |
| $FLMP_o$ | 0.83 | 0.80 | 0.78 | 0.86 | 0.82 | 0.81 |

vs. ordered priors). The smaller complexity of LIM is in line with previous attempts to measure the relative complexities of LIM and FLMP, such as the atheoretical simulation-based approach ([4] but see [5]), the semi-theoretical simulation-based approach [4], the theoretical simulation-based approach [2, 6, 22], and a direct computation of the GC [2].

The PPC's for all six designs considered are displayed in Table 1. It shows that the observations made for the $8 \times 2, n = 150$ design holds across the five remaining designs as well: LIM is simpler than FLMP; and models assuming ordered priors are simpler than models assuming uniform priors. Note that these conclusions would not have been possible based on PC or GC. For PC, all four models have the same complexity. GC, in contrast, would detect complexity differences between LIM and FLMP (i.e., the first conclusion), but due to its insensitivity to the parameter prior, the complexity differences between $\mathrm{LIM_u}$ and $\mathrm{LIM_o}$ on the one hand, and $\mathrm{FLMP_u}$ and $\mathrm{FLMP_o}$ on the other hand (i.e., the second conclusion) would have gone unnoticed.

## 5 Discussion

A theorist defining a model should clearly and explicitly specify at least the three following pieces of information: the model equation, the parameter prior range, and the parameter prior distribution. If any of these pieces is missing, the model should be regarded as incomplete, and therefore untestable. Consequently, any measure of generalizability should be sensitive to all three aspects of the model definition. Many currently popular generalizability measures do not satisfy this criterion, including AIC, BIC and MDL. A measure of generalizability that does take these three aspects of a model into account is the marginal likelihood [6, 7, 14, 23]. Often, the marginal likelihood is criticized exactly for its sensitivity to the prior range and distribution (e.g., [24]). However, in the light of the fact that the prior is a part of the model definition, I see the sensitivity of the marginal likelihood to the prior as an asset rather than a nuisance. It is precisely the measures of generalizability that are insensitive to the prior that miss an important aspect of the model.

Similarly, any stand alone measure of model complexity should be sensitive to all three aspects of the model definition, as all three aspects contribute to the model's complexity (with the model equation contributing two factors: the number of parameters and the functional form). Existing measures of complexity do not satisfy this requirement and are therefore incomplete. PC takes only part of the model equation into account, whereas GC takes only the model equation and the range into account. In contrast, the PPC currently proposed is sensitive to all these three aspects. It assesses model complexity using the predicted interval which contains all possible outcomes a model can generate. A narrow predicted interval (relative to the universal interval) indicates a simple model; a complex model is characterized by a wide predicted interval.

There is a tight coupling between the notions of information, knowledge and uncertainty, and the notion of model complexity. As parameters correspond to unknown variables, having more information available leads to fewer parameters and hence to a simpler model. Similarly, the more information there is available, the sharper the parameter prior, implying a simpler model. To put it differently, the less uncertainty present in a model, the narrower its predicted interval, and the simpler the model. For example, in model $M_b$, there is maximal uncertainty. Nothing but the range is known about $\theta$, so all values of $\theta$ are equally likely. In contrast, in model $M_f$, there is minimal uncertainty. In fact, $p_h$ is known for sure, so only a single value of $\theta$ is possible. This difference in uncertainty is translated in a difference in complexity. The same is true for the information integration models. Incorporating the order constraints in the priors reduces the uncertainty compared to the models without these constraints (it tells you, for example, that parameter $a_1$ is smaller than $a_2$). This reduction in uncertainty is reflected by a smaller complexity.

There are many different sources of prior information that can be translated in a range or distribution. The illustration using the information integration models highlighted that prior information can reflect meaningful information in the design. Alternatively, priors can be informed by previous applications of similar models in similar settings. Probably the purest form of priors are those that translate theoretical assumptions made by a model (see [16]). The fact that it is often difficult to formalize this prior information may not be used as an excuse to leave the prior unspecified. Sure it is a challenging task, but so is translating theoretical assumptions into the model equation. Formalizing theory, intuitions, and information is what model building is all about.

## Footnotes

*I am grateful to Michael Lee and Liz Bonawitz.

# References

[1] Myung, I. J. (2000) The importance of complexity in model selection. *Journal of Mathematical Psychology*, **44**, 190–204.

[2] Pitt, M. A., Myung, I. J., and Zhang, S. (2002) Toward a method of selecting among computational models of cognition. *Psychological Review*, **109**, 472–491.

[3] Shiffrin, R. M., Lee, M. D., Kim, W., and Wagenmakers, E. J. (2008) A survey of model evaluation approaches with a tutorial on hierarchical Bayesian methods. *Cognitive Science*, **32**, 1248–1284.

[4] Cutting, J. E., Bruno, N., Brady, N. P., and Moore, C. (1992) Selectivity, scope, and simplicity of models: A lesson from fitting judgments of perceived depth. *Journal of Experimental Psychology: General*, **121**, 364–381.

[5] Dunn, J. (2000) Model complexity: The fit to random data reconsidered. *Psychological Research*, **63**, 174–182.

[6] Myung, I. J. and Pitt, M. A. (1997) Applying Occam's razor in modeling cognition: A Bayesian approach. *Psychonomic Bulletin & Review*, **4**, 79–95.

[7] Vanpaemel, W. and Storms, G. (in press) Abstraction and model evaluation in category learning. *Behavior Research Methods*.

[8] Akaike, H. (1973) Information theory and an extension of the maximum likelihood principle. Petrov, B. and Csaki, B. (eds.), *Second International Symposium on Information Theory*, pp. 267–281, Academiai Kiado.

[9] Schwarz, G. (1978) Estimating the dimension of a model. *Annals of Statistics*, **6**, 461–464.

[10] Myung, I. J., Balasubramanian, V., and Pitt, M. A. (2000) Counting probability distributions: Differential geometry and model selection. *Proceedings of the National Academy of Sciences*, **97**, 11170–11175.

[11] Lee, M. D. (2002) Generating additive clustering models with minimal stochastic complexity. *Journal of Classification*, **19**, 69–85.

[12] Rissanen, J. (1996) Fisher information and stochastic complexity. *IEEE Transactions on Information Theory*, **42**, 40–47.

[13] Grünwald, P. (2000) Model selection based on minimum description length. *Journal of Mathematical Psychology*, **44**, 133–152.

[14] Lee, M. D. and Wagenmakers, E. J. (2005) Bayesian statistical inference in psychology: Comment on Trafimow (2003). *Psychological Review*, **112**, 662–668.

[15] Lee, M. D. and Vanpaemel, W. (2008) Exemplars, prototypes, similarities and rules in category representation: An example of hierarchical Bayesian analysis. *Cognitive Science*, **32**, 1403–1424.

[16] Vanpaemel, W. and Lee, M. D. (submitted) Using priors to formalize theory: Optimal attention and the generalized context model.

[17] Lee, M. D. (2008) Three case studies in the Bayesian analysis of cognitive models. *Psychonomic Bulletin & Review*, **15**, 1–15.

[18] Spiegelhalter, D., Thomas, A., Best, N., and Lunn, D. (2004) WinBUGS User Manual Version 2.0. *Medical Research Council Biostatistics Unit. Institute of Public Health, Cambridge*.

[19] Anderson, N. H. (1981) *Foundations of information integration theory*. Academic Press.

[20] Oden, G. C. and Massaro, D. W. (1978) Integration of featural information in speech perception. *Psychological Review*, **85**, 172–191.

[21] Massaro, D. W. (1998) *Perceiving Talking Faces: From Speech Perception to a Behavioral Principle*. MIT Press.

[22] Massaro, D. W., Cohen, M. M., Campbell, C. S., and Rodriguez, T. (2001) Bayes factor of model selection validates FLMP. *Psychonomic Bulletin and Review*, **8**, 1–17.

[23] Kass, R. E. and Raftery, A. E. (1995) Bayes factors. *Journal of the American Statistical Association*, **90**, 773–795.

[24] Liu, C. C. and Aitkin, M. (2008) Bayes factors: Prior sensitivity and model generalizability. *Journal of Mathematical Psychology*, **53**, 362–375.

